# Switch Packet Arbitration via Queue-Learning

**Timothy X Brown**
Electrical and Computer Engineering
Interdisciplinary Telecommunications
University of Colorado
Boulder, CO 80309-0530
timxb@colorado.edu

## Abstract

In packet switches, packets queue at switch inputs and contend for out-puts. The contention arbitration policy directly affects switch perfor-mance. The best policy depends on the current state of the switch and current traffic patterns. This problem is hard because the state space, possible transitions, and set of actions all grow exponentially with the size of the switch. We present a reinforcement learning formulation of the problem that decomposes the value function into many small inde-pendent value functions and enables an efficient action selection.

## 1   Introduction

Reinforcement learning (RL) has been applied to resource allocation problems in telecom-munications. e.g., channel allocation in wireless systems, network routing, and admis-sion control in telecommunication networks [1, 3, 7, 11]. These have demonstrated rein-forcement learning can find good policies that significantly increase the application reward within the dynamics of the telecommunications problems. However, a key issue is how to scale these problems when the state space grows quickly with problem size.

This paper focuses on packet arbitration for data packet switches. Packet switches are un-like telephone circuit switches in that packet transmissions are uncoordinated and clusters of traffic can simultaneously contend for switch resources. A *packet arbitrator* decides the order packets are sent through the switch in order to minimize packet queueing delays and the switch resources needed. Switch performance depends on the arbitration policy and the pattern of traffic entering the switch.

A number of packet arbitration strategies have been developed for switches. Many have fixed policies for sending packets that do not depend on the actual patterns of traffic in the network [10]. Under the worse case traffic, these arbitrators can perform quite poorly [8]. Theoretical work has shown consideration of future packet arrivals can have significant impact on the switch performance but is computationally intractable (NP-Hard) to use [4]. As we will show, a dynamic arbitration policy is difficult since the state space, possible transitions, and set of actions all grow exponentially with the size of the switch.

In this paper, we consider the problem of finding an arbitration policy that dynamically and efficiently adapts to traffic conditions. We present queue-learning, a formulation that effectively decomposes the problem into many small RL sub-problems. The independent

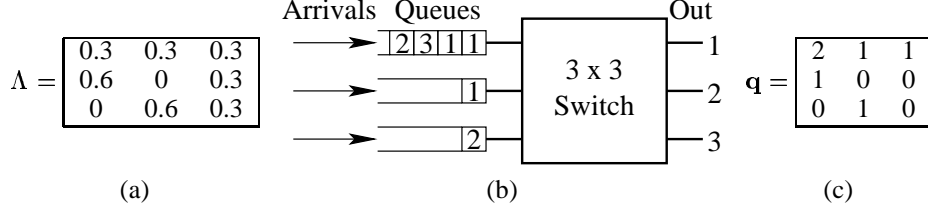

Figure 1: The packet arbitration model. (a) In each time slot, packet sources generate $\lambda_{ij} \in \Lambda$ packets on average at input $i$ for output $j$. (b) Packets arrive at an input-queued switch and are stored in queues. The number label on each packet indicates to which output the packet is destined. (c) The corresponding queue states, where $q_{ij} \in \mathbf{q}$ indicates the number of packets waiting at input $i$ destined for output $j$.

RL problems are coupled via an efficient algorithm that trades off actions in the different sub-problems. Results show significant performance improvements.

## 2 Problem Description

The problem is comprised of $N$ traffic sources generating traffic at each of $N$ inputs to a packet data switch as shown in Figure 1. Time is divided into discrete time slots and in each time slot each source generates 0 or 1 packets. Each packet that arrives at the input is labeled with which of the $N$ outputs the packet is headed. In every time slot, the switch takes packets from inputs and delivers them at their intended output. We describe the specific models for each used in this paper and then state the packet arbitration problem.

### 2.1 The Traffic Sources

At input $i$, a traffic source generates a packet destined for output $j$ with probability $\lambda_{ij}$ at the beginning of each time slot. If $\lambda_i^{in} = \sum_j \lambda_{ij}$ is the load on input $i$ and $\lambda_j^{out} = \sum_i \lambda_{ij}$ is the load on output $j$, then for stability we require $\lambda_i^{in} < 1.0 \ \forall i$ and $\lambda_j^{out} < 1.0 \ \forall j$.

The matrix $\Lambda = \{\lambda_{ij}\}$ only represents long term average loads between input $i$ and output $j$. We treat the case where packet arrivals are uncorrelated over time and between sources so that in each time slot, a packet arrives at input $i$ with probability $\lambda_i^{in}$ and given that we have an arrival, it is destined for output $j$ with probability $\lambda_{ij}/\lambda_i^{in}$. Let the set of packet arrivals be $\mathbf{e}$.

### 2.2 The Switch

The switch alternates between accepting newly arriving packets and sending packets in every time slot. At the start of the time slot the switch sends packets waiting in the input queues and delivers them to the correct output where they are sent on. Let $\mathbf{a} = \{a_{ij}\}$ represent the set of packets sent where $a_{ij} = 1$ if a packet is sent from input $i$ to output $j$ and $a_{ij} = 0$ otherwise. The packets it can send are limited by the input and output constraints: the switch can send at most one packet per input and can deliver at most one packet to each output. After sending packets, the new arrivals are added at the input and the switch moves to the next time slot. Other switches are possible, but this is the simplest and a common architecture in high-speed switches.

### 2.3 The Input Queues

Because the traffic sources are un-coordinated, it is possible for multiple packets to arrive in one time slot at different inputs, but destined for the same output. Because of the output constraint, only one such packet may be sent and the others buffered in queues, one queue per input. Thus packet queueing is unavoidable and the goal is to limit the delays due to queueing.

The queues are random access which means packets can be sent in any order from a queue. For the purposes of this paper, all packets waiting at an input and destined for the same output are considered equivalent. Let $\mathbf{q} = \{q_{ij}\}$ be a matrix where $q_{ij}$ is the number of packets waiting at input $i$ for output $j$ as shown in Figure 1c.

### 2.4 Packet Arbitration

The packet arbitration problem is: Given the state of the input queues, $\mathbf{q}$, choose a set of packets to send, $\mathbf{a}$, so at most one packet is sent from each input and at most one packet is delivered to each output. We want a packet arbitration policy that minimizes the expected packet wait time.

When $\mathbf{a}$ is sent the remaining packets must wait at least one more time slot before they can be sent. Let $|\mathbf{q}|$ be the total number of packets in all the input queues, let $|\mathbf{e}|$ be the number of new arrivals, and let $|\mathbf{a}|$ be the number of packets sent. Thus, the total wait of all packets is increased by the number of packets that remain: $|\mathbf{q}| + |\mathbf{e}| - |\mathbf{a}|$. By Little's theorem, the expected wait time is proportional to the expected number of packets waiting in each time slot [10]). Thus, we want a policy that minimizes the expected value of $|\mathbf{q}| + |\mathbf{e}| - |\mathbf{a}|$.

The complexity of this problem is high. Given an $N$ input and $N$ output switch. The input and output constraints are met with equality if $\mathbf{a}$ is a subset of a permutation matrix (zeros everywhere except that every row has at most one one and every column has one one). This implies there are as many as $N!$ possible $\mathbf{a}$ to choose from. In each time slot at each input, a packet can arrive for one of $N$ outputs or not at all. This implies as many as $(N + 1)^N$ possible transitions after each send. If each $q_{ij}$ ranges from 0 to $b$ packets, then the number of states in the system is $b^{N^2}$. A minimal representation would only indicate whether each sub-queue is empty or not, resulting in $2^{N^2}$ states. Thus, every aspect of the problem grows exponentially in the size of the switch.

Traditionally switching solves these problems by not considering the possible next arrivals, and using a search algorithm with time-complexity polynomial in $N$ that considers only the current state $\mathbf{q}$. For instance the problem can be formulated as a so-called matching problem and polynomial algorithms exist that will send the largest $\mathbf{a}$ possible [2, 6, 8].

While maximizing the packets sent in every time slot may seem like a solution, the problem is more interesting than this. In general, many possible $\mathbf{a}$ will maximize the number of packets that are sent. Which one can we send now so that we will be in the best possible state for future time slots? Some heuristics can guide this choice, but these are insensitive to the traffic pattern $\Lambda$ [9]. Further, it can be shown that to minimize the total wait it may be necessary to send *less* than the maximum number of packets in the current time slot [4]. So, we look to a solution that efficiently finds policies that minimize the total wait by adapting to the current traffic pattern.

The problem is especially amenable to RL for two reasons. (1) Packet rates are fast, up to millions of packets per second so that many training examples are available. (2) Occasional bad decisions are not catastrophic. They only increase packet delays somewhat, and so it is possible to freely learn in an online system. The next section describes our solution.

# 3 Queue-Learning Solution

At any given time slot, $t$, the system is in a particular state, $\mathbf{q}_t$. New packets, $\mathbf{e}_t$, arrive and the packet arbitrator can choose to send any valid $\mathbf{a}_t$. The cost, $c(\mathbf{q}, \mathbf{e}, \mathbf{a})$ is the $(|\mathbf{q}_t| + |\mathbf{e}_t| - |\mathbf{a}_t|)$ packets that remain. The task of the learner is to determine a packet arbitration policy that minimizes the total average cost. We use the Tauberian approximation, that is, we assume the discount factor is close enough to 1 so that the discounted reward policy is equivalent to the average reward policy [5]. Since minimizing the expected value of this cost is equivalent to minimizing the expected wait time, this formulation provides an exact match between RL and the problem task.

As shown already every aspect of this problem scales badly. The solution to this problem is three fold. First we use online learning and afterstates [12] to eliminate the need to average over the $(N+1)^N$ possible next states. Second, we show how the value function can yield a set of inputs into a polynomial algorithm for choosing actions. Third, we decompose the value function so the effective number of states is much smaller than $b^{N^2}$. We describe each in turn.

## 3.1 Afterstates

RL methods solve MDP problems by learning good approximations to the optimal value function, $J^*$. A single time slot consists of two stages: new arrivals are added to the queues and then packets are sent (see Figure 2). The value function could be computed after either of these stages. We compute it after packets are sent since we can use the notion of afterstates to choose the action. Since the packet sending process is deterministic, we know the state following the send action. In this case, the Bellman equation is:

$$J^*(\mathbf{q}) = E_{\mathbf{e}} \left\{ \min_{\mathbf{a} \in A(\mathbf{q}, \mathbf{e})} \left[ c(\mathbf{q}, \mathbf{e}, \mathbf{a}) + \gamma J^*(\mathbf{q}') \right] \right\}$$

where $A(\mathbf{q}, \mathbf{e})$ is the set of actions available in the current state $\mathbf{q}$ after arrival event $\mathbf{e}$, $c(\mathbf{q}, \mathbf{e}, \mathbf{a}) = |\mathbf{q}| + |\mathbf{e}| - |\mathbf{a}|$ is the effective immediate cost, $\gamma$ is the discount factor, and $E_{\mathbf{e}}\{\cdot\}$ is the expectation over possible events and the resulting next state is $\mathbf{q}'$.

We learn an approximation to $J^*$ using TD(0) learning. At time-step $t$ on a transition from state $\mathbf{q}_t$ to $\mathbf{q}_{t+1}$ on action $\mathbf{a}_t$ after event $\mathbf{e}_t$, we update an estimate to $J$ via

$$J(\mathbf{q}_t) = J(\mathbf{q}_t) + \alpha_t \left[ c(\mathbf{q}_t, \mathbf{a}_t, \mathbf{e}_t) + \gamma J(\mathbf{q}_{t+1}) - J(\mathbf{q}_t) \right]$$

where $\alpha_t$ is the learning step size.

With afterstates, the action (which set of packets to send) depends on both the current state and the event. The best action is the one that results in the lowest value function in the next state (which is known deterministically given $\mathbf{q}_t$, $\mathbf{e}_t$, and $\mathbf{a}_t$). In this way, afterstated eliminates the need to average over a large number of non-zero transitions to find the best action.

## 3.2 Choosing the Action

We compare every action with the action of not sending any packets. The best action, is the set of packets meeting the input and output constraints that will reduce the value function the most compared to not sending any packets.

Each input-output pair $(i, j)$ has an associated queue at the input, $q_{ij}$. Packets in $q_{ij}$ contend with other packets at input $i$ and other packets destined for output $j$. If we send a packet from $q_{ij}$, then no packet at the same input or output will be sent. In other words, packets at

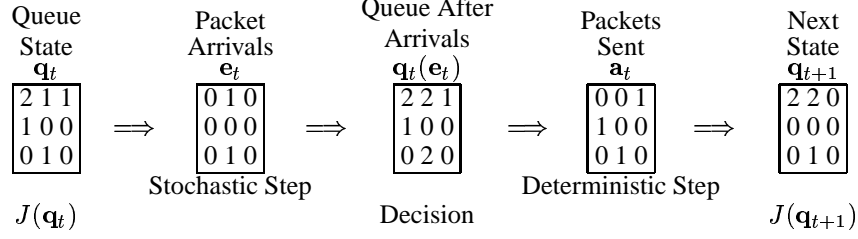

| Queue State $\mathbf{q}_t$ | | Packet Arrivals $\mathbf{e}_t$ | | Queue After Arrivals $\mathbf{q}_t(\mathbf{e}_t)$ | | Packets Sent $\mathbf{a}_t$ | | Next State $\mathbf{q}_{t+1}$ |

$$\begin{matrix} 2\ 1\ 1 \\ 1\ 0\ 0 \\ 0\ 1\ 0 \end{matrix} \implies \begin{matrix} 0\ 1\ 0 \\ 0\ 0\ 0 \\ 0\ 1\ 0 \end{matrix} \implies \begin{matrix} 2\ 2\ 1 \\ 1\ 0\ 0 \\ 0\ 2\ 0 \end{matrix} \implies \begin{matrix} 0\ 0\ 1 \\ 1\ 0\ 0 \\ 0\ 1\ 0 \end{matrix} \implies \begin{matrix} 2\ 2\ 0 \\ 0\ 0\ 0 \\ 0\ 1\ 0 \end{matrix}$$

Stochastic Step  ·  Deterministic Step

$J(\mathbf{q}_t)$  ·  Decision  ·  $J(\mathbf{q}_{t+1})$

Figure 2: Timing of packet arrivals and sends relative to decisions and the value function.

$q_{ij}$ interact primarily with packets in the same row and column. Packets in other rows and columns only have an indirect effect on the value of sending a packet from $q_{ij}$.

This suggests the following approximation. Let $\mathbf{q}(\mathbf{e})$ be the number of packets in every subqueue after arrivals $\mathbf{e}$ in state $\mathbf{q}$ and before the decision. Let $\delta_{ij}(\mathbf{q}(\mathbf{e}))$ be the reduction in the value function if one packet is sent from subqueue $(i,j)$ ($\delta_{ij}(\mathbf{q}(\mathbf{e})) = 0$ if the subqueue is empty). We can reformulate the best action as:

$$\mathbf{a} = \mathrm{argmax}_{\mathbf{a}'=\{a_{ij}\}} \sum_i \sum_j \delta_{ij}(\mathbf{q}(\mathbf{e}))a_{ij}$$

subject to the constraints:

$$\begin{aligned} a_{ij} &\in \{0,1\} \quad \forall i,j \\ \sum_i a_{ij} &\leq 1 \quad \forall j \\ \sum_j a_{ij} &\leq 1 \quad \forall i \end{aligned}$$

This problem can be solved as a linear program and is also known as the weighted matching or the assignment problem which has a polynomial time solution [13]. In this way, we reduce the search over the $O(N!)$ possible actions to a polynomial time solution.

### 3.3 Decomposing the Value Function

The interaction between queues in the same row or the same column is captured primarily by the input and output constraints. This suggests a further simplifying approximation with the following decomposition.

We compute a separate value function for each $q_{ij}$, denoted $J_{ij}(\mathbf{q})$. In principle, this can depend on the entire state $\mathbf{q}$, but can be reduced to consider only elements of the state relevant to $q_{ij}$. Every $q_{ij}$ estimates its associated value function $J_{ij}(\mathbf{q})$ based on the packets at input $i$ and packets destined for output $j$.

Many forms of $J_{ij}(\mathbf{q})$ could be considered, but we consider a linear approximation. Let $q_i^{in}$ be the total number of packets waiting at input $i$. Let $q_j^{out}$ be the total number of packets waiting for output $j$.

With these variables we define a linear approximation with parameters $\theta = (\theta_0, \dots, \theta_6)$:

$$J_{ij}(\mathbf{q}) = \theta_0 + \theta_1 q_{ij} + \theta_2 q_{ij}^2 + \theta_3 q_i^{in} + \theta_4 (q_i^{in})^2 + \theta_5 q_j^{out} + \theta_6 (q_j^{out})^2 \qquad (1)$$

It follows the value of sending a packet (compared to not sending a packet) from $q_{ij}$ is

$$\delta_{ij}(\mathbf{q}(\mathbf{e})) = \theta_1 + \theta_3 + \theta_5 + \theta_2(2q_{ij}(\mathbf{e}) - 1) + \theta_4(2q_i^{in}(\mathbf{e}) - 1) + \theta_6(2q_j^{out}(\mathbf{e}) - 1).$$

This is computed for each $(i, j)$ and used in the weighted matching of Section 3.2 to compute which packets to send. Learning for this problem is standard TD(0) for linear approximations [12]. The combination of decomposition and linear value function approximation reduces the problem to estimating $O(N^2)$ parameters.

No explicit exploration is used since from the perspective of $J_{ij}$, enough stochasticity already exists in the packet arrival and send processes. To assist the switch early in the learning, the switch sends the packets from a maximum matching in each time slot (instead of the packets selected by queue learning). This initial assist period during the training was found to bring the switch into a good operating regime from which it could learn a better policy.

In summary, we simplify the exponential computation for this problem by decomposing the state into $N^2$ substates. Each substate computes the value of sending a packet versus not sending a packet, and a polynomial algorithm computes the action that maximizes the total value across substates subject to the input and output constraints.

## 4 Implementation Issues

A typical high speed link rate is at OC-3 rates (155Mbps). In ATM at this rate, the packet rate is 366k time slots/s or less than 30 sec for $10^7$ time slots. For learning, the number of floating point operations per time slot is approximately $4PN^2$ where $P$ is the number of parameters in the linear approximation. At the above packet rate, for an $N = 8$ switch, this translates into 650 MFLOPS which is within existing highend microprocessor capacity. For computation of the packets to send, the cost is approximately $PN^2$ to compute the weights. To compute the maximum weight matching an $O(N^3 \log N)$ algorithm exists [13].

New optical transport technologies are pushing data rates one and two orders of magnitude greater than OC-3 rates. In this case, if computing is limited then the queue-learning can learn on a subsample of time slots. To compute the packets to send, the decomposition has a natural parallel implementation that can divide it among processors. Massively parallel neural networks can also be used to compute the maximum weighted matching [2, 9].

## 5 Simulation Results

We applied our procedure to $8 \times 8$ switches under different loads. The parameters used in the experiment are shown in Table 1. In each experiment, the queue-learning was trained for an initial period, and then the mean wait time, $W_{ql}$ is measured over a test period. We compared performance to two alternatives. One alternative sends the largest number of packets in every time slot. If multiple sets are equally large it chooses randomly between them. We simulate this arbitrator and measure the mean packet wait time, $W_{sendmax}$. The best possible switch is a so-called output-queued switch [10]. Such a switch is difficult to build at high-speeds, but we can compute its mean packet wait time, $W_{out}$, via simulation. The results are specified in normalized form as $(W_{sendmax} - W_{ql})/(W_{sendmax} - W_{out})$. Thus if our queue-learning solution is no better than a max send arbitrator, the gain will be 0 and if we achieve the performance of the output-queued switch, the gain will be 1.

We experimented on five different traffic loads. $\Lambda_1$ is a uniform load of 0.6 packets per input per time slot with each packet uniformly destined for one of the outputs. Similarly, $\Lambda_2$ is a uniform load of 0.9. The uniform load is a common baseline scenario for evaluating switches.

$\Lambda_3$ and $\Lambda_4$ are random matrices where the sum of loads per row and column are 0.6 and 0.9 (as in $\Lambda_1$ and $\Lambda_2$) but the distribution is not uniform. This is generated by summing $N$ permutation matrices and than scaling the entries to yield the desired row and column sums

Table 1: RL parameters.

| Parameter | Value |
|---|---|
| Discount, $\gamma$ | 0.99 |
| Learn Rate, $\alpha_t$ | $\frac{0.01}{1+t/100000}$ |
| Assist Period | $10^6$ time slots |
| Train Period | $10^7$ time slots |
| Test Period | $10^7$ time slots |

Table 2: Simulation Results.

| Switch Loading | Normalized Wait Reduction ($\pm 2\%$) |
|---|---|
| $\Lambda_1$ (uniform 0.6 load) | 10% |
| $\Lambda_2$ (uniform 0.9 load) | 50% |
| $\Lambda_3$ (random 0.6 load) | 14% |
| $\Lambda_4$ (random 0.9 load) | 70% |
| $\Lambda_5$ (truncated 0.9 load) | 84% |

(e.g. Figure 1a). The random load is a more realistic in that loads tend to vary among the different input/output pairs.

$\Lambda_5$ is $\Lambda_4$, except that all $\lambda_{ij}$ for the last $N/2$ outputs is set to zero. This simulates the more typical case of traffic being concentrated on a few outputs.

We emphasize that a different policy is learned for each of these loads. The different loads suggest the kinds of improvements that we might expect if queue-learning is implemented. The results for the five loads are given in Table 2.

## 6 Conclusion

This paper showed that queue learning is able to learn a policy that significantly reduces the wait times of packets in a high-speed switch. It uses a novel decomposition of the value function combined with efficient computation of the action to overcome the problems a traditional RL approach would have with the large number of states, actions, and transitions. This is able to gain 10% to 84% of the possible reductions in wait times. The largest gains are when the network is more heavily loaded and delays are largest. The gains are also largest when the switch load is least uniform which is what is most likely to be encountered in practice.

Traditional thinking in switching is that input-queued switches are much worse than the optimal output-queued switches and improving performance would require increasing switching speeds (the electronic switching is already the slowest part of the otherwise optical networking), or using information of future arrivals (which may not exists and in any case is NP-Hard to use optimally). The queue-learning approach is able to use its estimates of the future impact of its packet send decisions in a consistent framework that is able to bridge the majority of the gap between current input queueing and optimal output queueing.

**Acknowledgment**

This work was supported by CAREER Award: NCR-9624791.

# References

[1] Boyan, J.A., Littman, M.L., "Packet routing in dynamically changing networks: a reinforcement learning approach," in Cowan, J.D., et al., ed. *Advances in NIPS 6*, Morgan Kauffman, SF, 1994. pp. 671–678.

[2] Brown, T.X, Lui, K.H., "Neural Network Design of a Banyan Network Controller," *IEEE JSAC*, v. 8, n. 8, pp. 1428–1438, Oct., 1990.

[3] Brown, T.X, Tong, H., Singh, S., "Optimizing admission control while ensuring quality of service in multimedia networks via reinforcement learning," in *Advances NIPS 11*, ed. M. Kearns et al., MIT Press, 1999.

[4] Brown, T.X, Gabow, H.N., "Future Information in Input Queueing," submitted to *Computer Networks*, April 2001.

[5] Gabor, Z., Kalmar, Z., Szepesvari, C., "Multi-criteria Reinforcement Learning," *International Conference on Machine Learning*, Madison, WI, July, 1998.

[6] J. Hopcroft and R. Karp, "An $n^{5/2}$ algorithm for maximum matchings in bipartite graphs", SIAM J. Computing 2, 4, 1973, pp 225-231.

[7] Marbach, P., Mihatsch, M., Tsitsiklis, J.N., "Call admission control and routing in integrated service networks using neuro-dynamic programming," *IEEE J. Selected Areas in Comm.*, v. 18, n. 2, pp. 197–208, Feb. 2000.

[8] McKeown, N., Anantharam, V., Walrand, J., "Achieving 100% Throughput in an Input-Queued Switch," *Proc. of IEEE INFOCOM '96*, San Francisco, March 1996.

[9] Park, Y.-K., Lee, G., "NN Based ATM Scheduling with Queue Length Based Priority Scheme," *IEEE J. Selected Areas in Comm.*, v. 15, n. 2 pp. 261–270, Feb. 1997.

[10] Pattavina, A., *Switching Theory: Architecture and Performance in Broadband ATM Networks,* John Wiley and Sons, New York, 1998.

[11] Singh, S.P., Bertsekas, D.P., "Reinforcement learning for dynamic channel allocation in cellular telephone systems," in *Advances in NIPS 9*, ed. Mozer, M., et al., MIT Press, 1997. pp. 974–980.

[12] Sutton, R.S., Barto, A.G., *Reinforcement Learning: an Introduction,* MIT Press, 1998.

[13] Tarjan, R.E., *Data Structures and Network Algorithms*, Soc. for Industrial and Applied Mathematics, Philidelphia, 1983.
